# Simplifying Constraint Inference with Inverse Reinforcement Learning

**Adriana Hugessen**
Mila, Université de Montréal

**Harley Wiltzer**
Mila, McGill University

**Glen Berseth**
Mila, Université de Montréal

`{adriana.knatchbull-hugessen,wiltzerh,glen.berseth}@mila.quebec`

## Abstract

Learning safe policies has presented a longstanding challenge for the reinforcement learning (RL) community. Various formulations of safe RL have been proposed; However, fundamentally, tabula rasa RL must learn safety constraints through experience, which is problematic for real-world applications. Imitation learning is often preferred in real-world settings because the experts' safety preferences are embedded in the data the agent imitates. However, imitation learning is limited in its extensibility to new tasks, which can only be learned by providing the agent with expert trajectories. For safety-critical applications with sub-optimal or inexact expert data, it would be preferable to learn only the safety aspects of the policy through imitation, while still allowing for task learning with RL. The field of inverse constrained RL, which seeks to infer constraints from expert data, is a promising step in this direction. However, prior work in this area has relied on complex tri-level optimizations in order to infer safe behavior (constraints). This challenging optimization landscape leads to sub-optimal performance on several benchmark tasks. In this work, we present a simplified version of constraint inference that performs as well or better than prior work across a collection of continuous-control benchmarks. Moreover, besides improving performance, this simplified framework is easier to implement, tune, and more readily lends itself to various extensions, such as offline constraint inference. Our code is made available at https://github.com/ahugs/simple-icrl.

## 1  Introduction

Reinforcement learning (RL) has made significant advances in recent years, yet real-world applications of RL remain limited due to various challenges, including particularly safety concerns [Dulac-Arnold et al., 2021]. One common setting where RL holds promise for real-world deployments is replacing existing, possibly sub-optimal, human-managed control policies. For example, in the field of power network control, decisions are often made by a combination of automatic control policies combined with careful human monitoring and intervention [Marot et al., 2022]. However, concerns regarding the deployment of autonomous systems on safety-critical tasks have hindered the adoption of RL for replacing legacy control systems. This is particularly true for deep RL, which functions as a black box controller.

In the setting of replacing legacy controllers, however, the challenge is simplified due to access to data produced by the current control system. While existing control policies may be sub-optimal, they are nonetheless "safe" in the sense that they obey some explicitly or implicitly defined constraints corresponding to a human understanding of safety. Ideally, we would like to be able to learn an RL policy that can outperform the current system, in terms of optimizing the reward, while continuing to respect these constraints. When these constraints are unknown (i.e. defined implicitly through human

actions), the implicit safety procedures that are being followed may not be explicitly known or have an obvious encoding. Thus, it would be beneficial to infer the constraints directly from trajectories produced by the existing control policy.

Typically, in settings with large-scale offline data, imitation learning [Schaal, 1996] or offline RL [Kumar et al., 2020, Kostrikov et al., 2022] might be used. However, these methods only extract a single policy that satisfies constraints and optimizes a reward function. Moreover, imitation learning generally cannot outperform the expert, which is prohibiting in cases where we wish to improve over current control policies. Offline RL can learn policies that outperform the demonstration [Kumar et al., 2020], however, offline RL cannot ensure that safety constraints remain satisfied without access to the constraint function or constraint violation annotations in the dataset.

Ideally, we would like to extract safety constraints from the data based on the expert behavior, which can then be used downstream to constrain task-specific learning. Learning constraints from expert trajectories is the purview of the field of constraint inference. Various methods have been proposed in this domain, with early work focusing on simple settings such as tabular MDPs [Scobee and Sastry, 2020]. More generally, inverse constrained reinforcement learning can infer constraints in continuous settings using parameterized constraint functions and adapting IRL methods such as maximum entropy IRL [Ziebart et al., 2008] to the constrained setting by solving a constrained MDP in the inner optimization. However, this tri-level optimization creates a challenging landscape for constraint inference.

In this work, we demonstrate that the constrained MDP inner loop is an unnecessary complication and that regular IRL techniques can recover as good and sometimes better solutions to constraint inference problems than these more complicated methods. This result is significant because it allows us to simplify the training dynamics and complexity of constraint inference methods and implies that advances in sub-domains of IRL can be directly applied to the constraint inference case. For example, recent progress in offline IRL [Yue et al., 2023, Kim et al., 2023] can be readily adapted to the constraint inference case to infer constraints entirely offline, significantly increasing the scope of applicability in real-world settings.

In particular, we make the following contributions. First, we show that inverse constrained RL and inverse RL are equivalent under certain classes of constraint functions. Next, we experimentally validate this claim. Finally, we propose some practical modifications to adapt IRL to constraint inference tasks and conduct ablations over these algorithmic choices to understand how to improve the stability and performance of constraint inference.

## 2   Related Work

**General imitation learning**   The problem of imitation learning (IL) is primarily concerned with learning a policy that produces similar behavior to a class of reference policies. Specifically, given transition data from "experts", the goal is to produce a new policy that, in a sense, generates a similar transition distribution. The simplest approach to IL is behavior cloning (BC) [Pomerleau, 1988], which estimates a policy via maximum likelihood to match the conditional distribution on actions conditioned on state observations. BC tends to perform well in the regime of massive data but suffers from compounding errors over long trajectories as state observations veer outside the coverage of the dataset [Ross et al., 2011]. Methods such as DAgger [Ross et al., 2011] show that, given the ability to interact with the environment during IL and to query expert trajectories, strong policies can be learned with relatively few queries to the expert. More recently, behavior cloning losses have been integrated into standard reinforcement learning methods to enhance *offline RL* performance — that is, RL from a fixed (offline) dataset [Fujimoto and Gu, 2021]. Alternative approaches to IL have focused on learning a policy that induces a similar state-visitation distribution to the dataset [Pirotta et al., 2023]. Such an objective explicitly accounts for the long-term behavior of a policy as opposed to the myopic predictions of BC methods, resulting in more coherent learned behaviors over longer horizons with less data. IL methods as discussed above can generally learn performant imitation policies, but provide no insight about the reward function that the expert is implicitly optimizing; this is the focus of inverse reinforcement learning, which will be useful for the purpose of learning constraint functions.

**Inverse Reinforcement Learning**   *Inverse reinforcement learning* (IRL) learns a policy that can mimic expert trajectories and also aims to infer a reward function that explains the expert behavior [Ng et al., 2006]. The IRL problem is considerably more difficult than the IL problem; in particular, there is no unique reward function that maximally explains the expert behavior. The work of Abbeel and Ng [2004] alleviates this issue with a *maximum margin constraint* in the optimization. The work of Ziebart et al. [2008] resolves this problem by inferring the most likely reward function under a particular probabilistic model; this approach is widely known as *maximum entropy IRL* (MaxEnt IRL). One approach for achieving strong guarantees in IRL is to approximate the expert occupancy distribution *in the operator norm* — that is, to have a low discrepancy between expected returns under any reward function. Achieving this generally involves a bilevel optimization procedure where one alternates between optimizing a policy for a given reward function, and then adversarially training the reward function [Gleave and Toyer, 2022]. The work of Garg et al. [2021] reduces this problem in the maximum entropy framework by estimating a soft action-value function ($Q$), inferring the reward function from a novel "inverse Bellman operator" applied to $Q$ and showing that the policy $\pi(\cdot \mid s) \propto \exp(Q(s, \cdot))$ is MaxEnt-optimal. Effectively, their method reduces the bilevel optimization discussed above into a simpler RL optimization. Alternatively, Swamy et al. [2021] assumes that the reward function lies in a reproducing kernel Hilbert space, which allows them to obtain an unbiased estimate of the adversarial reward function from data for any policy – this is yet another method for eliminating one of the layers of the bilevel optimization discussed above. However, these aforementioned IRL methods have not considered the issue of modeling environmental constraints which can be transferred when learning policies for novel tasks.

**Inverse Constrained Reinforcement Learning**   Early work in constraint inference focused primarily on particular settings such as convex constraints [Menner et al., 2019, Miryoosefi et al., 2019], or tabular RL [Scobee and Sastry, 2020, McPherson et al., 2021, Chou et al., 2020]. More recent methods have integrated the power of deep learning within the constraint inference framework through adaptions of MaxEnt IRL to the constrained setting through inverse constrained reinforcement learning (ICRL) [Malik et al., 2021, Liu et al., 2023a, Kim et al., 2023]. These methods essentially replace the forward RL inner loop of IRL with a constrained version of the problem by casting it as a constrained MDP (CMDP) and solving by Lagrangian or other methods. This adds additional complexity to the IRL problem which is already difficult due to the bi-level optimization and identifiability issues [Gleave and Toyer, 2022]. Though these prior works have sometimes considered simple IRL baselines [Malik et al., 2021, Liu et al., 2023a], none have specifically outlined the connections between IRL and ICRL, nor have they attempted to optimize IRL for constraint inference. A related line of work from the control theory community also considers learning constraints from expert data as control barrier functions, [Robey et al., 2020, Castaneda et al., 2023, Lindemann et al., 2024], however these methods rely on access to a known or learned model of the dynamics, which we do not assume.

## 3   Background

**Imitation Learning.**   Imitation learning is the process of training a new policy to reproduce expert policy behavior. Given a set of expert trajectories $D_e = \{\tau_e^{(i)}\}_{i=0}^N$, which follow a policy $\pi_E$, a new policy $\pi$ can be learned to match these trajectories using supervised learning, i.e. maximizing the expectation: $\mathbb{E}_{\pi_E}\left[\sum_{t=0}^T \log \pi(a_t|s_t, \theta)\right]$. While this simple method can work well, it often suffers from distribution mismatch issues leading to compounding errors as the learned policy deviates from the expert's behavior [Ross et al., 2011]. Inverse reinforcement learning avoids this issue by extracting a reward function from observed optimal behavior [Ng et al., 2000].

**Maximum Entropy IRL**   As discussed in Section 2 there are several formulations of IRL. Here we describe maximum entropy IRL [Ziebart et al., 2008, Ziebart, 2010]. Given a set of expert trajectories $D_e = \{\tau_e^{(i)}\}_{i=0}^N$, which follow a policy $\pi_E$, a policy $\pi$ can be trained to produce similar trajectories by solving the min-max problem

$$\min_{r \in \mathcal{R}} \max_{\pi} \left( \mathbb{E}_\pi \left[ \sum_{t=0}^T r(s_t, a_t) \right] + H(\pi) - \mathbb{E}_{\pi_E} \left[ \sum_{t=0}^T r(s_t, a_t) \right] \right), \tag{1}$$

where $r$ is a learned reward function and $H(\pi)$ is a causal entropy term. Hence, IRL is searching for a reward function $r$ that is high for the expert $\pi_E$ and low for other policies.

**Constrained RL** One way to formulate constrained MDPs (CMDPs) is through trajectory-level constraints. In this formulation, the agent must solve the following objective

$$\max_{\pi} \mathbb{E}_{\pi} \left[ \sum_{t=0}^{T} r(s_t, a_t) \right] \text{ s.t. } \mathbb{E}_{\pi} \left[ \sum_{t=0}^{T} c(s_t, a_t) \right] \leq \delta, \tag{2}$$

where $c(s_t, a_t)$ is a cost function with $\delta$ budget.

The most common method for solving this problem is Lagrangian variations of RL methods, such as SAC-Lag [Ha et al., 2020], where Equation (2) is converted to a min-max problem

$$\min_{\lambda \geq 0} \max_{\pi} \mathbb{E}_{\pi} \left[ \sum_{t=0}^{T} \gamma^t r(s_t, a_t) \right] - \lambda \left( \mathbb{E}_{\pi} \left[ \sum_{t=0}^{T} \gamma^t c(s_t, a_t) \right] - \delta \right). \tag{3}$$

**Inverse constrained RL** Inverse constrained reinforcement learning (ICRL) is a class of methods used to infer constraints from expert trajectories. As in imitation learning and IRL, it is assumed that we have access to a set of expert trajectories $D_e = \{\tau_e^{(i)}\}_{i=0}^{N}$, which follow a policy $\pi_E$, however, it is now assumed that the expert policy is optimal under a CMDP instead of an MDP. It is generally assumed that the agent has access to the unconstrained (nominal) MDP with which to interact, and that the (constraint-agnostic) reward is observed by the agent when interacting with the environment.

Kim et al. [2023] present a general formulation of ICRL as game-solving, though we note that their formulation was preceded by that of Malik et al. [2021], and, though motivated differently, generally matches it, up to some implementation details.

As shown in prior work [Swamy et al., 2023], the inverse RL problem can be cast as a two-player zero-sum game

$$\mathsf{Opt}_{\mathsf{IRL}}(\Pi, \mathcal{F}) = \min_{\pi \in \Pi} \sup_{f \in \mathcal{F}} J(\pi_E, f) - J(\pi, f) \tag{4}$$

where $\mathcal{F}$ is convex and compact and $J(\pi, f) = \mathbb{E}_{\pi} \left[ \sum_{t=0}^{T} f(s_t, a_t) \right]$.

Similarly, the constrained RL problem in Equation (2) with a single constraint can also be cast as a two-player zero-sum game

$$\min_{\pi \in \Pi} \max_{\lambda \geq 0} -J(\pi, r) + \lambda(J(\pi, c) - \delta). \tag{5}$$

Finally, Kim et al. [2023] show that for inverse constrained RL, these two games can be combined into a three-player game of the following form (where $r$ is a given reward function in a class $\mathcal{F}_r$)

$$\mathsf{Opt}_{\mathsf{ICRL}}(\Pi, \mathcal{F}_r, \mathcal{F}_c) = \sup_{c \in \mathcal{F}_c} \max_{\lambda > 0} \min_{\pi \in \Pi} J(\pi_E, r - \lambda c) - J(\pi, r - \lambda c). \tag{6}$$

Practically speaking, solving this game involves running IRL where the inner loop optimization solves a constrained MDP using a Lagrangian version of RL, such as SAC-Lag [Ha et al., 2020].

## 4 Method

Starting from Equation (6), we now demonstrate that this tri-level optimization is equivalent to a simpler bi-level optimization under certain classes of constraint functions and that this bi-level optimization is equivalent to the IRL game-solving formulation in Equation (4).

### 4.1 Inverse constrained RL as IRL

Note that there are two outer maximizations in Equation (6), the first over constraints in the class $\mathcal{F}_c$ and the second over $\lambda > 0$. Our key insight is that, under a broad class of constraint functions $\mathcal{F}_c$, this tri-level optimization can be cast as a bi-level optimization, reducing ICRL to IRL.

**Theorem 4.1** *Let $\pi_E \in \Pi$ and a reward function $r \in \mathcal{F}_r$ be given. Consider the objective*

$$\mathsf{Opt}_{\mathsf{S-ICRL}}(\Pi, \mathcal{F}_r, \mathcal{F}_c) = \max_{c \in \mathcal{F}_c} \min_{\pi \in \Pi} J(\pi_E, r - c) - J(\pi, r - c). \tag{7}$$

Then, if $\mathcal{F}_c$ is a convex cone (that is, $c \in \mathcal{F}_c \implies \lambda c \in \mathcal{F}_c$ for any $\lambda \geq 0$), it holds that $\mathsf{Opt}_{\mathsf{ICRL}}(\Pi, \mathcal{F}_r, \mathcal{F}_c) = \mathsf{Opt}_{\mathsf{S-ICRL}}(\Pi, \mathcal{F}_r, \mathcal{F}_c)$.

*Moreover, suppose $\Pi$ is compact and $\mathcal{F}_r = \mathcal{F}_c = \mathcal{F}$ is a vector space with elements $f : (s, a) \mapsto \langle \phi(s, a), w_f \rangle$, where $\phi : \mathcal{S} \times \mathcal{A} \to [0, 1]^d$ is a fixed (not necessarily known) feature map and $w_f \in \mathbb{R}^d$ identifies the elements of $\mathcal{F}$. Then any solution to $\mathsf{Opt}_{\mathsf{IRL}}(\Pi, \mathcal{F} - r)$[1] is a solution to $\mathsf{Opt}_{\mathsf{ICRL}}(\Pi, \mathcal{F}, \mathcal{F})$—that is, the ICRL problem is simply an instance of IRL.*

*Proof.* See Appendix B. □

The intuition for this is the following. The only difference between solving a constrained MDP with Lagrange multiplier $\lambda$ and solving an unconstrained MDP with a cost penalty term weighted by $\lambda$ is that, in the former case $\lambda$ is optimized to ensure constraint satisfaction. Indeed, given the optimal $\lambda^*$ a priori, constrained MDPs can be solved exactly by optimizing the later problem. This distinction is generally important when the cost function is fixed and unknown, however in the case of inverse learning where we *learn* the cost function from a class closed to scalar multiplication, it is possible to learn a cost that *directly* ensures constraint satisfaction, without scaling by the Lagrange multiplier. Note that optimizing over a convex cone $\mathcal{F}_c$ can pose challenges when exact optimization is required—for instance, convex cones violate a compactness condition required to prove regret bounds for inverse constrained RL given by Kim et al. [2023]. However, in large scale applications that *approximately* optimize a constraint model represented by a deep neural network, one is often *already* in the setting where these regret bounds do not hold. Hence, when employing parameterized constraint inference through deep learning, it is sufficient to select an appropriate class as the output activation of our neural network and use a bi-level optimization to learn the scaled constraint.

## 4.2 Techniques for stabilizing constraint learning

Given that IRL and ICRL are mathematically equivalent under the class of constraint functions described above, the question of how best to perform constraint inference through inverse learning becomes largely a *practical* one. Is the optimization landscape smoother if we explicitly optimize for $\lambda$ as a Lagrange multiplier or implicitly as part of the constraint function? Do previous implementations of Lagrangian methods add additional algorithmic components that could explain the improved performance of ICRL over IRL in prior work? Moreover, are there *other* regularizations or modifications that we could consider? In this regard, we suggest the following practical modifications to IRL for the constraint learning case, which we will test in Section 5.

**Bounding rewards**    One advantage of prior methods' use of a binary classifier for constraint representations is that it restricts learned costs within a range of positive values [Malik et al., 2021]. In practice, it is advantageous for interpretability and transferability to restrict the constraint function to be strictly positive, so that the learned constraint can only discourage the agent from visiting certain states not visited by the expert. Bounding the rewards above can also be beneficial for preventing divergences during training. However, in the case of IRL for constraint inference, it is necessary to optimize over a convex cone and hence a binary activation function cannot be used.

We propose and evaluate two possible solutions. As a baseline solution, we consider simply clipping the output values of a linear activation function to within some fixed positive range, while also re-scaling rewards using the rolling mean and variance. Though clipping technically violates the convex cone assumption, in practice the reward scaling should restrict the range of the constraint function necessary to ensure constraint satisfaction.

This baseline solution is potentially problematic, however. Besides violating the convex cone assumption, hard clipping the output of a neural network can be challenging due to the potential for vanishing gradients. Alternatively, we propose softly enforcing the constraint function to the positive range by using a Leaky ReLU activation and adding an L2 regularization term to the imitation gap loss, similar to Malik et al. [2021], so that the loss becomes

$$\mathcal{L}(\pi, c) = J(\pi_E, r - c) - J(\pi, r - c) + \mathbb{E}_{s,a \sim \tau_E}\big[c(s, a)^2\big], \tag{8}$$

where $\mathbb{E}_{s,a \sim \tau_E}\big[c(s, a)^2\big]$ is an L2 regularization on the learned cost of the expert trajectories. Unlike in Malik et al. [2021], we only regularize the output of the constraint function on the expert data,

allowing the constraint function to grow unconstrained, *except* on states visited by the expert, where the cost should be zero by the assumption that the expert trajectories are safe, i.e. non-violating.

**IRL with separate critics**   One component of prior methods for constraint inference that is implicitly added when utilizing a Lagrangian method for policy optimization, is the use of separate critics for the reward and constraint functions. Separate critics may potentially be beneficial for learning, since separating the critics disentangles the value function learning of the reward (which is not changing during training) with the constraint function (which changes as the constraint is learnt). By Q-decomposition [Russell and Zimdars, 2003], additive rewards can also be evaluated by separate additive critics and hence it is a straightforward modification to implement separate critics in any actor-critic IRL implementation without the additional Lagrangian optimization, which we outline in Algorithm 1.

---

**Algorithm 1** IRL for ICRL - Separate Critics

---

1: Initialize cost network parameters $\phi$, policy parameters $\psi$ and cost and reward critic parameters $\theta$ and $\alpha$. Let $M$ be the number of cost function updates and $N$ be the number of policy updates.
2: **for** $m = 0, 1 \ldots, \text{M}$ **do**
3:    **for** $n = 0, \ldots, N$ **do**
4:       Collect experience $(s_t, a_t, s_{t-1}, r_t)$ from nominal environment and add to buffer $\beta$
5:       Sample batch $\{(s_i, a_i, s_{i+1}, r_i)\}_{i=1}^{B} \sim \beta$.
6:       Compute learned cost for batch $c_i = R_\phi(s_i, a_i) \;\; \forall i \in B$
7:       Update cost critic parameters $\theta$ from $c_i$ and reward critic parameters $\alpha$ from $r_i$.
8:       Update policy parameters using critic $Q(s, a) = Q_\alpha(s, a) - Q_\theta(s, a)$
9:    **end for**
10:    Update cost function using loss $\mathbb{E}_{\pi_E}[R_\phi(s, a)] - \mathbb{E}_{\pi_\psi}[R_\phi(s, a)]$
11: **end for**

---

**Last-layer policy resetting**   Recent work has demonstrated that the primacy bias caused by plasticity loss can be a significant challenge for training deep RL agents [Nikishin et al., 2022]. This challenge is exacerbated in the case of IRL and particularly ICRL since the learned cost function changes constantly during training while the true rewards are provided from the environment, potentially biasing early policy training towards learning the unconstrained policy. To combat this plasticity loss issue, we adopt the recommendations from Lyle et al. [2023] for MLPs, which advises periodically resetting the final layer of the network during training.

## 5   Experiments

In this section, we conduct a series of experiments across several environments in order to answer the following questions: (1) How does IRL perform on constraint inference tasks compared to Lagrangian methods? and (2) How do the proposed modifications or regularizations over vanilla IRL improve performance on constraint inference tasks?

**Environments**   For our experiments, we consider the virtual environments for benchmarking inverse constraint learning, introduced by Liu et al. [2023a] since these were specially designed to test the performance of constraint inference tasks and also provide a recent baseline for Lagrangian-based constraint inference methods, including expert data. The environments include five MuJoCo environments, *Ant*, *Half Cheetah*, *Walker 2D*, *Swimmer* and *Inverted Pendulum*, modified to include constraints, which are primarily binary restrictions on the x-position of the agent. For more details, see Liu et al. [2023a].

**Evaluation metrics**   Evaluating constrained RL is challenging due to the dual objectives of optimizing rewards and satisfying constraints, which are typically in conflict with one another. Hence, in our evaluation, we report two metrics, following Liu et al. [2023a], *feasible rewards* and *violation rate*. Feasible rewards are calculated as the total returns for an episode up to the point of the first constraint violation. The violation rate is the percentage of episodes with one or more constraint violations. Using these metrics allows us to compare the baselines along the same axis as prior work.

We compare all methods according to average performance in the last 50 testing episodes and report statistics (IQM, Median, Mean and Optimality Gap) with bootstrapped 95% confidence intervals computed across five seeds according to the method recommended in Agarwal et al. [2021]. We provide the full learning curves in Appendix C.2.

**Prior methods for comparison**  We compare our results against two baselines from Liu et al. [2023a]. The first is their implementation of MaxEnt ICRL [Malik et al., 2021] (denoted **MECL**), which is the canonical method for Lagrangian-based constraint inference that provides the primary baseline for our method. The second is their implementation of GAIL for constraint inference (denoted **GACL**), introduced as a baseline for **MECL** in Malik et al. [2021]. This is a basic version of IRL for constraint inference, which uses the GAIL algorithm [Ho and Ermon, 2016] with an additive log term for the constraint. In Liu et al. [2023a], they test an additional two algorithms which, for conciseness, we do not include in this work, as one has been shown to underperform other methods in previous work and the other introduces Bayesian inference into the constraint learning, which is not considered here. We obtain results for **MECL** and **GACL** using the implementation and hyperparameters provided by the authors of Liu et al. [2023a]. We note that for our implementation of constraint inference, we use SAC as the forward RL algorithm, similar to Kim et al. [2023], whereas Malik et al. [2021] and Liu et al. [2023a] use PPO with an entropy regularization term. [2].

**Experimental Setup**  As mentioned, we use SAC for policy optimization. As the IRL algorithm, we utilize a version of maximum entropy IRL as implemented in Zeng et al. [2022]. The learnt constraint function is parameterized as a two-layer MLP with linear output activation. We adopt the hyperparameters for SAC as used in Achiam [2018], except that we use automatic $\alpha$-tuning [Haarnoja et al., 2018]. All the hyperparameters for IRL were set to those used in Zeng et al. [2022], except the learning rate on the constraint function which we tune for each environment. We find that when training with separate critics, warm-starting the policy learning by training without the constraint critic initially is beneficial, so we warm-start configurations with separate critics with 500k environment steps. We train all variations in all environments for 5M environment steps across five seeds. Full hyperparameter configurations are included in Appendix A.1.[3]

In the following sections, the modifications proposed in Section 4.2 are labeled as **IRL-Base** for IRL with clipping and no modifications, **L2** for using L2 regularization in place of clipping, **SC** for adding separate critics and **PR** for policy reset. **IRL-Plus** includes all proposed modifications (**L2**, **SC**, **PR**).

## 5.1 IRL versus ICRL

First, we compare our IRL implementation to the baseline methods. In Figure 1 we present a summary of our findings across environments, showing the final expert-normalized performance, aggregated across all five environments, with 95% bootstrapped confidence intervals computed across five seeds.

Notably, basic IRL (**IRL-Base**) already performs quite favorably versus **MECL**. Statistics on the feasible rewards are generally improved for **IRL-Base** versus **MECL**, though this improvement does not always have statistical significance. On the other hand, violation rate is improved over **MECL** with statistical significance across all but one statistic. Notably, the base IRL method performs similarly to the prior baseline IRL method **GACL**, with no statistical difference in feasible rewards across all four statistics, though it generally outperforms in terms of the violation rate.

With the added modifications, **IRL-Plus** outperforms **MECL** with statistical significance across all four metrics in feasible rewards and all but one metric in violation rate. **IRL-Plus** also outperforms the baseline IRL method **GACL** with statistical significance across almost all metrics for both feasible rewards and violation rate, with the median being the only exception.

### 5.1.1 Sub-optimal Expert Trajectories

Previous work has found that constraint inference with a Lagrangian inner loop is more robust to suboptimal expert trajectories as compared to previous IRL methods which cannot learn effectively [Liu et al., 2023a]. We conduct experiments on the suboptimal dataset for *Half Cheetah* provided in

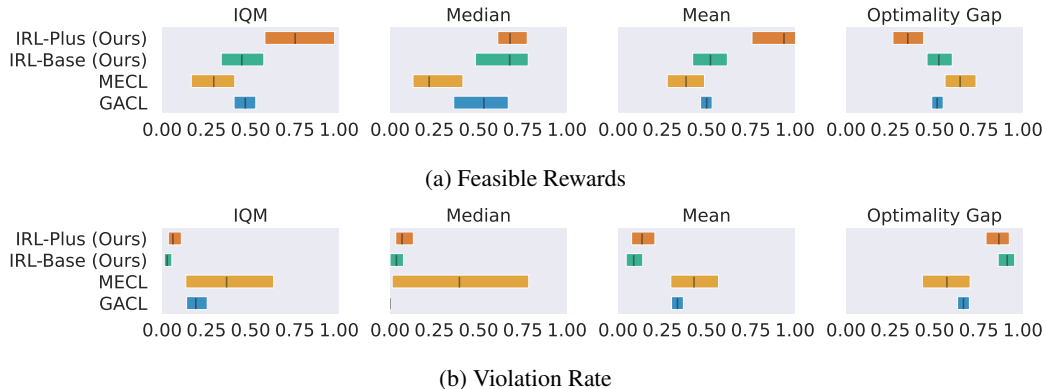

(a) Feasible Rewards

(b) Violation Rate

Figure 1: Normalized final performance aggregated across the five MuJoCo environments, where feasible rewards are normalized by the expert returns. Both **IRL-Base** and **IRL-Plus** perform better than the baseline ICRL method **MECL** on average, with **IRL-Plus** out-performing by a statistically significant margin across most metrics, for both feasible rewards and violation rate.

Liu et al. [2023a] to test whether this remains true with our implementation. The sub-optimal datasets are constructed to contain varying percentages of unsafe expert trajectories, i.e. expert trajectories with one or more constraint violations. For these experiments, we test the combination of all proposed modifications (**IRL-Plus**), i.e. L2 regularization on the expert rewards, separate critics and policy resets versus the two baselines **MECL** and **GACL**.

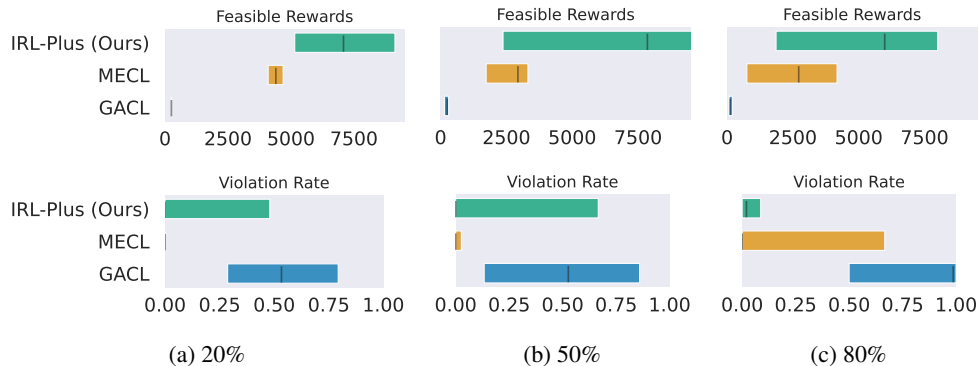

(a) 20%     (b) 50%     (c) 80%

Figure 2: Interquartile mean (IQM) of final performance on *Half-Cheetah* with sub-optimal trajectories in ratios of 20%, 50% and 80%. In all three scenarios, **IRL-Plus** out-performs **MECL** in terms of feasible rewards, though only with statistical significance in the 20% scenario. Neither **MECL** nor **IRL-Plus** clearly outperforms in terms of violation rate, with both methods achieving near zero.

As shown in Figure 2, in all cases, our method performs much better than the prior IRL baseline method **GACL**, which does not achieve any feasible rewards. Our method also generally outperforms the Lagrangian-based method **MECL** though with notably high variance, so that the improved performance is only statistically significant in the 20% scenario. Overall, however, it is clear that the ICRL method **MECL** does not outperform **IRL-Plus** on a statistically significant basis.

## 5.2 Impact of Modifications

Here, we more closely examine the impact of the proposed modifications (Section 4.2) on constraint inference with IRL. Overall, while the impact of the modifications varies somewhat across environments, as can be seen in Figure 4, aggregated across all environments on a normalized basis, the combination of all modifications, i.e. L2 regularization, separate critics and policy resets (**IRL-Plus**) results in an increase in feasible rewards vs **IRL-Base** which is statistically significant in all statistics

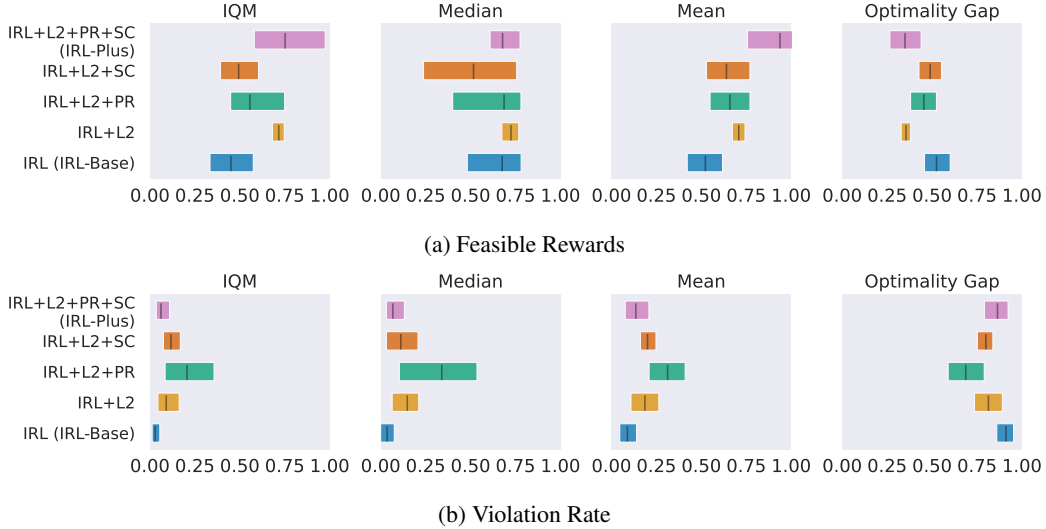

(a) Feasible Rewards

(b) Violation Rate

Figure 3: Normalized final performance for all ablations aggregated across the five MuJoCo environments, where feasible rewards are normalized by the expert returns. IRL with L2-regularization, separate critics and policy resets (**IRL-Plus**) generally performs the best in terms of feasible rewards with only a slight increase in violation rate versus **IRL-Base**.

except the median, with only a slight increase in the violation rate which is not statistically significant in any of the statistics.

**L2 regularization versus clipping with reward normalization**   Overall, the most substantial increase in performance comes from the L2 regularization versus clipping with reward normalization. This modification improves feasible rewards by a statistically significant amount as measured by three of the four metrics, excepting the median. At the same time, it results in a small increase in violation rate, however this increase is not statistically significant for any of the four statistics. The addition of L2 regularization has the largest impact on *Walker2D* as can be seen in Figure 4.

**Policy reset and separate critics**   Adding both separate critics and policy reset (**SC+PR**) slightly improves performance over **IRL+L2**, across most statistics, in both feasible rewards and violation rate, though the impact is marginal and only statistically significant in some metrics. Interestingly, neither the separate critics modification nor the policy reset modification alone results in an improvement over **IRL+L2**, suggesting it is the interaction of these two modifications that are beneficial. For example, separate critics without policy resets (**IRL+L2+SC**) slightly hurts performance in terms of feasible rewards with similar violation rate, and policy resets without separate critics (**IRL+L2+PR**) slightly increases violation rate without much improvement in feasible rewards, though we note that these effects are generally not statistically significant. Notably, when combining separate critics with policy resets we reset the last layer of only the constraint critic and not the reward critic. This allows the reward critic to learn the unchanging reward function without resets, while the constraint critic can adapt more quickly through resets to the changing constraint function, which may explain the performance boost observed when combining these two methods.

While the impact of separate critics and policy resets is marginal overall, in certain environments such as *Inverted Pendulum* and *Half Cheetah*, it does confer a meaningful improvement in feasible rewards, as can be seen in Figure 4, though with notably high variance across seeds.

## 6    Discussion and Conclusions

Overall, we have demonstrated that Lagrangian-based methods for constraint inference are theoretically equivalent to IRL for a broad constraint class. Moreover, these Lagrangian methods do not appear to confer performance benefits over simpler IRL methods, when IRL is implemented using MaxEnt IRL with constraint functions parameterized by neural networks with appropriate activation

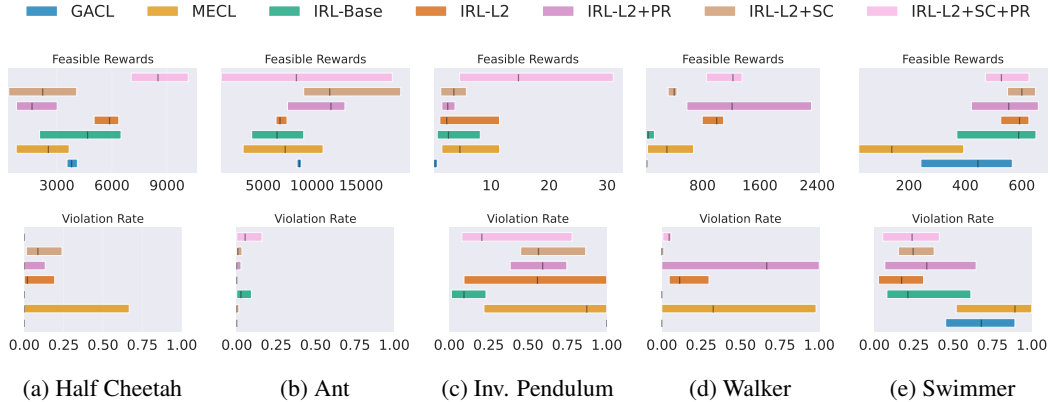

Figure 4: Interquartile mean (IQM) of final performance for all proposed modifications and baselines in each individual MuJoCo environment. All methods display relatively high variance including **MECL**. Even so, in three of the five environments, at least one method achieves CI that do not overlap with the baseline ICRL method **MECL** in feasible returns while being within the CI of the baseline in violation rate. In the remaining two environments, confidence intervals for both feasible rewards and violation rate overlap between the baseline ICRL method **MECL**.

functions. Further, with the addition of some simple-to-implement practical modifications to the IRL algorithm we can further improve performance of IRL over previous Lagrangian-based ICRL methods for constraint inference tasks.

**Advantages of IRL over ICRL**    Besides performance improvements, using IRL for constraint inference confers several benefits to the field. IRL reduces the complexity of the optimization problem versus ICRL by removing a level in the nested optimization, making a simpler to implement solution that can more easily be applied to new domains. For example, we highlight that we did not have to perform any hyperparameter tuning, except on the constraint function learning rate, to achieve decent results in all environments. In contrast, previous ICRL methods use many environment-specific hyperparameters [Liu et al., 2023a]. Additionally, the use of IRL methods to solve constraint inference problems could facilitate applying the much broader field of research in IRL and its sub-fields to various extensions of constraint inference that have not yet been fully explored. These could include, in particular, offline constraint inference using offline IRL techniques [Yue et al., 2023].

Importantly, we emphasize that the constraint function learned through our modified IRL procedure is equivalent to the constraint function learned through Lagrangian-based ICRL methods, up to a scalar multiplicative constant, and hence can be treated as a constraint function in downstream transfer tasks. More specifically, though the constraint function is *learned* as an additive term in the reward of an unconstrained MDP, the learned function can be *transferred* as a constraint in a constrained MDP. Solving this constrained MDP will require scaling the learned constraint by a new Lagrangian term when optimizing for the new transfer task. Hence, there is no loss in the transferability of the constraint when performing constraint inference with IRL versus ICRL.

**Limitations and Future Work**    While we show the feasibility of IRL for constraint inference as an alternative to ICRL, we note that there is substantial variance in the solutions produced by IRL on a per environment basis, as can be observed in Figure 4. Reducing this variance should be a priority of future work. Moreover, ongoing work is still needed to improve the performance of constraint inference across more challenging environments such as *Inverted Pendulum*.

Additionally, while our results indicate that our **IRL-Plus** method outperforms previous ICRL method **MECL**, we note that there are several algorithmic differences between our implementation and that of Liu et al. [2023a], such as the use of SAC vs PPO as the RL policy optimization algorithm. While we believe we have effectively demonstrated certain implementations of IRL can be competitive with ICRL we cannot conclude that IRL is strictly better than ICRL in all cases.

As mentioned, an exciting extension for future work would be to explore various subdomains of IRL in the context of constraint inference, particularly the offline IRL case, and to apply these techniques to real datasets.

## Acknowledgments and Disclosure of Funding

We want to acknowledge funding support from NSERC, FRQNT and CIFAR, as well as compute support from Digital Research Alliance of Canada, Mila IDT, and NVidia.

## Footnotes

[1]The notation $\mathcal{F} - r$ refers to the set $\{f - r : f \in \mathcal{F}\}$.

[2]Though we have attempted to re-implement Malik et al. [2021] using SAC, our performance is low compared to Liu et al. [2023a]. We include full results of our SAC implementation in Appendix C.1

[3]Our code is available at: https://github.com/ahugs/simple-icrl

## References

P. Abbeel and A. Y. Ng. Apprenticeship learning via inverse reinforcement learning. In *Proceedings of the twenty-first international conference on Machine learning*, page 1, 2004.

J. Achiam. Spinning Up in Deep Reinforcement Learning. 2018.

R. Agarwal, M. Schwarzer, P. S. Castro, A. C. Courville, and M. Bellemare. Deep reinforcement learning at the edge of the statistical precipice. *Advances in neural information processing systems*, 34:29304–29320, 2021.

A. Barreto, W. Dabney, R. Munos, J. J. Hunt, T. Schaul, H. P. van Hasselt, and D. Silver. Successor features for transfer in reinforcement learning. *Advances in neural information processing systems*, 30, 2017.

F. Castaneda, H. Nishimura, R. T. McAllister, K. Sreenath, and A. Gaidon. In-distribution barrier functions: Self-supervised policy filters that avoid out-of-distribution states. In *Learning for Dynamics and Control Conference*, pages 286–299. PMLR, 2023.

G. Chou, D. Berenson, and N. Ozay. Learning constraints from demonstrations. In *Algorithmic Foundations of Robotics XIII: Proceedings of the 13th Workshop on the Algorithmic Foundations of Robotics 13*, pages 228–245. Springer, 2020.

R. Dadashi, A. A. Taiga, N. Le Roux, D. Schuurmans, and M. G. Bellemare. The value function polytope in reinforcement learning. In *International Conference on Machine Learning*, pages 1486–1495. PMLR, 2019.

G. Dulac-Arnold, N. Levine, D. J. Mankowitz, J. Li, C. Paduraru, S. Gowal, and T. Hester. Challenges of real-world reinforcement learning: definitions, benchmarks and analysis. *Machine Learning*, 110(9):2419–2468, 2021.

S. Fujimoto and S. S. Gu. A minimalist approach to offline reinforcement learning. *Advances in neural information processing systems*, 34:20132–20145, 2021.

D. Garg, S. Chakraborty, C. Cundy, J. Song, and S. Ermon. Iq-learn: Inverse soft-q learning for imitation. *Advances in Neural Information Processing Systems*, 34:4028–4039, 2021.

A. Gleave and S. Toyer. A primer on maximum causal entropy inverse reinforcement learning. *arXiv preprint arXiv:2203.11409*, 2022.

S. Ha, P. Xu, Z. Tan, S. Levine, and J. Tan. Learning to walk in the real world with minimal human effort. *arXiv preprint arXiv:2002.08550*, 2020.

T. Haarnoja, A. Zhou, K. Hartikainen, G. Tucker, S. Ha, J. Tan, V. Kumar, H. Zhu, A. Gupta, P. Abbeel, et al. Soft actor-critic algorithms and applications. *arXiv preprint arXiv:1812.05905*, 2018.

J. Ho and S. Ermon. Generative adversarial imitation learning. *Advances in neural information processing systems*, 29, 2016.

K. Kim, G. Swamy, Z. Liu, D. Zhao, S. Choudhury, and S. Z. Wu. Learning shared safety constraints from multi-task demonstrations. *Advances in Neural Information Processing Systems*, 36, 2023.

I. Kostrikov, A. Nair, and S. Levine. Offline reinforcement learning with implicit q-learning. In *The Tenth International Conference on Learning Representations, ICLR 2022, Virtual Event, April 25-29, 2022*. OpenReview.net, 2022. URL `https://openreview.net/forum?id=68n2s9ZJWF8`.

A. Kumar, A. Zhou, G. Tucker, and S. Levine. Conservative q-learning for offline reinforcement learning. *Advances in Neural Information Processing Systems*, 33:1179–1191, 2020.

L. Lindemann, A. Robey, L. Jiang, S. Das, S. Tu, and N. Matni. Learning robust output control barrier functions from safe expert demonstrations. *IEEE Open Journal of Control Systems*, 2024.

G. Liu, Y. Luo, A. Gaurav, K. Rezaee, and P. Poupart. Benchmarking constraint inference in inverse reinforcement learning. In *The Eleventh International Conference on Learning Representations, ICLR 2023, Kigali, Rwanda, May 1-5, 2023*. OpenReview.net, 2023a. URL https://openreview.net/pdf?id=vINj_Hv9szL.

Z. Liu, Z. Guo, H. Lin, Y. Yao, J. Zhu, Z. Cen, H. Hu, W. Yu, T. Zhang, J. Tan, et al. Datasets and benchmarks for offline safe reinforcement learning. *arXiv preprint arXiv:2306.09303*, 2023b.

C. Lyle, Z. Zheng, E. Nikishin, B. A. Pires, R. Pascanu, and W. Dabney. Understanding plasticity in neural networks. In *International Conference on Machine Learning*, pages 23190–23211. PMLR, 2023.

S. Malik, U. Anwar, A. Aghasi, and A. Ahmed. Inverse constrained reinforcement learning. In *International conference on machine learning*, pages 7390–7399. PMLR, 2021.

A. Marot, A. Kelly, M. Naglic, V. Barbesant, J. Cremer, A. Stefanov, and J. Viebahn. Perspectives on future power system control centers for energy transition. *Journal of Modern Power Systems and Clean Energy*, 10(2):328–344, 2022. doi: 10.35833/MPCE.2021.000673.

D. L. McPherson, K. C. Stocking, and S. S. Sastry. Maximum likelihood constraint inference from stochastic demonstrations. In *2021 IEEE Conference on Control Technology and Applications (CCTA)*, pages 1208–1213. IEEE, 2021.

M. Menner, P. Worsnop, and M. N. Zeilinger. Constrained inverse optimal control with application to a human manipulation task. *IEEE Transactions on Control Systems Technology*, 29(2):826–834, 2019.

S. Miryoosefi, K. Brantley, H. Daume III, M. Dudik, and R. E. Schapire. Reinforcement learning with convex constraints. *Advances in Neural Information Processing Systems*, 32, 2019.

A. Y. Ng, S. Russell, et al. Algorithms for inverse reinforcement learning. In *Icml*, volume 1, page 2, 2000.

A. Y. Ng, A. Coates, M. Diel, V. Ganapathi, J. Schulte, B. Tse, E. Berger, and E. Liang. Autonomous inverted helicopter flight via reinforcement learning. In *Experimental robotics IX: The 9th international symposium on experimental robotics*, pages 363–372. Springer, 2006.

E. Nikishin, M. Schwarzer, P. D'Oro, P.-L. Bacon, and A. Courville. The primacy bias in deep reinforcement learning. In *International conference on machine learning*, pages 16828–16847. PMLR, 2022.

M. Pirotta, A. Tirinzoni, A. Touati, A. Lazaric, and Y. Ollivier. Fast imitation via behavior foundation models. In *The Twelfth International Conference on Learning Representations*, 2023.

D. A. Pomerleau. Alvinn: An autonomous land vehicle in a neural network. *Advances in neural information processing systems*, 1, 1988.

A. Robey, H. Hu, L. Lindemann, H. Zhang, D. V. Dimarogonas, S. Tu, and N. Matni. Learning control barrier functions from expert demonstrations. In *2020 59th IEEE Conference on Decision and Control (CDC)*, pages 3717–3724. IEEE, 2020.

S. Ross, G. Gordon, and D. Bagnell. A reduction of imitation learning and structured prediction to no-regret online learning. In *Proceedings of the fourteenth international conference on artificial intelligence and statistics*, pages 627–635. JMLR Workshop and Conference Proceedings, 2011.

S. J. Russell and A. Zimdars. Q-decomposition for reinforcement learning agents. In *Proceedings of the 20th international conference on machine learning (ICML-03)*, pages 656–663, 2003.

S. Schaal. Learning from demonstration. *Advances in neural information processing systems*, 9, 1996.

D. R. R. Scobee and S. S. Sastry. Maximum likelihood constraint inference for inverse reinforcement learning. In *8th International Conference on Learning Representations, ICLR 2020, Addis Ababa, Ethiopia, April 26-30, 2020*. OpenReview.net, 2020. URL `https://openreview.net/forum?id=BJliakStvH`.

G. Swamy, S. Choudhury, J. A. Bagnell, and S. Wu. Of moments and matching: A game-theoretic framework for closing the imitation gap. In *International Conference on Machine Learning*, pages 10022–10032. PMLR, 2021.

G. Swamy, D. Wu, S. Choudhury, D. Bagnell, and S. Wu. Inverse reinforcement learning without reinforcement learning. In *International Conference on Machine Learning*, pages 33299–33318. PMLR, 2023.

J. Weng, H. Chen, D. Yan, K. You, A. Duburcq, M. Zhang, Y. Su, H. Su, and J. Zhu. Tianshou: A highly modularized deep reinforcement learning library. *Journal of Machine Learning Research*, 23(267):1–6, 2022. URL `http://jmlr.org/papers/v23/21-1127.html`.

S. Yue, G. Wang, W. Shao, Z. Zhang, S. Lin, J. Ren, and J. Zhang. Clare: Conservative model-based reward learning for offline inverse reinforcement learning. In *International Conference on Learning Representations (ICLR)*, 2023.

S. Zeng, C. Li, A. Garcia, and M. Hong. Maximum-likelihood inverse reinforcement learning with finite-time guarantees. *Advances in Neural Information Processing Systems*, 35:10122–10135, 2022.

B. D. Ziebart. *Modeling purposeful adaptive behavior with the principle of maximum causal entropy*. Carnegie Mellon University, 2010.

B. D. Ziebart, A. L. Maas, J. A. Bagnell, A. K. Dey, et al. Maximum entropy inverse reinforcement learning. In *Aaai*, volume 8, pages 1433–1438. Chicago, IL, USA, 2008.

# A  Experiment Details

## A.1  Hyperparameters

All of our code is based on the Tianshou [Weng et al., 2022] and FSRL [Liu et al., 2023b] implementations of SAC and SAC-Lagrangian, respectively. Here we include all the hyperparameter configurations for our experiments. Any hyperparameters not listed here use the default hyperparameters in their respective libraries (Tianshou version 1.0.0 and FSRL version 0.1.0).

| SAC Parameters | |
| --- | --- |
| $\tau$ | 0.005 |
| $\gamma$ | 0.99 |
| $\alpha$ LR | 0.0003 |
| $\alpha$ optimizer | Adam |
| Critic hidden layers | [256, 256] |
| Critic optimizer | Adam |
| Critic learning rate | 0.001 |
| Actor hidden layers | [256, 256] |
| Actor learning rate | 0.001 |
| Actor optimizer | Adam |
| n-step returns | 1 |

| Training Parameters | |
| --- | --- |
| Max epochs | 1000 |
| Batch size | 128 |
| Environment steps per epoch | 5000 |
| Gradient steps per environment step | 1 |
| Test episodes per epoch | 5 |
| Episodes between gradient updates | 1 |
| Buffer size | 1000000 |

| Constraint Learning Parameters | IRL-Base | IRL-L2 |
| --- | --- | --- |
| Output clip range | [0,20] | None |
| Output activation | Linear | Leaky ReLU |
| Hidden layers | [64, 64] | [64, 64] |
| Optimizer | Adam | Adam |
| Weight decay | 0.001 | 0.001 |
| Batch size | 5000 | 5000 |
| Gradient steps per epoch | 1 | 1 |
| L2 regularization coefficient | 0 | 0.1 |

The constraint function learning rate was tuned over two values (0.001, 0.0001) per environment with the following final parameters:

| Half Cheetah | Ant | Inverted Pendulum | Walker | Swimmer |
| --- | --- | --- | --- | --- |
| 0.0001 | 0.0001 | 0.001 | 0.0001 | 0.001 |

## A.2  Computational Resources

Each run (consisting of five seeds) was trained on a node with a single GPU (varying GPU resources were used), 6 CPUs and 6GB of RAM per CPU. A single run across three seeds under this configuration took approximately 1.5 days to complete training.

# B  Proof of Theorem 4.1

We prove both claims individually.

## B.1 Step 1: $\mathsf{Opt}_{\mathsf{ICRL}} = \mathsf{Opt}_{\mathsf{S-ICRL}}$

We begin by proving the first claim, that $\mathsf{Opt}_{\mathsf{ICRL}}(\Pi, \mathcal{F}_r, \mathcal{F}_c) = \mathsf{Opt}_{\mathsf{S-ICRL}}(\Pi, \mathcal{F}_r, \mathcal{F}_c)$ when $\mathcal{F}_c$ is a convex cone. Firstly, it is simple to show that $\mathsf{Opt}_{\mathsf{ICRL}}(\Pi, \mathcal{F}_r, \mathcal{F}_c) \geq \mathsf{Opt}_{\mathsf{S-ICRL}}(\Pi, \mathcal{F}_r, \mathcal{F}_c)$,

$$
\begin{aligned}
\mathsf{Opt}_{\mathsf{ICRL}}(\Pi, \mathcal{F}_r, \mathcal{F}_c) &= \max_{c \in \mathcal{F}_c} \max_{\lambda \geq 0} \min_{\pi \in \Pi} J(\pi_E, r - \lambda c) - J(\pi, r - \lambda c) \\
&\geq \max_{c \in \mathcal{F}_c} \min_{\pi \in \Pi} J(\pi_E, r - 1 \cdot c) - J(\pi, r - 1 \cdot c) \\
&= \mathsf{Opt}_{\mathsf{S-ICRL}}(\Pi, \mathcal{F}_r, \mathcal{F}_c).
\end{aligned}
$$

It remains to show that $\mathsf{Opt}_{\mathsf{ICRL}}(\Pi, \mathcal{F}_r, \mathcal{F}_c) \leq \mathsf{Opt}_{\mathsf{S-ICRL}}(\Pi, \mathcal{F}_r, \mathcal{F}_c)$. We have

$$
\begin{aligned}
\mathsf{Opt}_{\mathsf{S-ICRL}}(\Pi, \mathcal{F}_r, \mathcal{F}_c) &= \max_{c \in \mathcal{F}_c} \min_{\pi \in \Pi} J(\pi_E, r - c) - J(\pi, r - c) \\
&\geq \max_{c \in \mathcal{F}_c} \min_{\pi \in \Pi} J(\pi_E, r - \lambda c) - J(\pi, r - \lambda c) \qquad \forall \lambda \geq 0 \\
&\geq \max_{c \in \mathcal{F}_c} \max_{\lambda \geq 0} \min_{\pi \in \Pi} J(\pi_E, r - \lambda c) - J(\pi, r - \lambda c) \\
&= \mathsf{Opt}_{\mathsf{ICRL}}(\Pi, \mathcal{F}_r, \mathcal{F}_c),
\end{aligned}
$$

where the first inequality holds since $\lambda c \in \mathcal{F}_c$ whenever $c \in \mathcal{F}_c$ by the hypothesis that $\mathcal{F}_c$ is a convex cone. It follows that $\mathsf{Opt}_{\mathsf{ICRL}}(\Pi, \mathcal{F}_r, \mathcal{F}_c) = \mathsf{Opt}_{\mathsf{S-ICRL}}(\Pi, \mathcal{F}_r, \mathcal{F}_c)$. ∎

## B.2 Step 2: IRL Solves ICRL

We now prove the second claim. For any policy $\pi$, we define the *successor features* [Ng et al., 2000, Ziebart et al., 2008, Barreto et al., 2017] $\psi^\pi : \mathcal{S} \to \mathbb{R}^d$ according to

$$
\psi^\pi(s) = \mathbb{E}\left[ \sum_{t \geq 0} \gamma^t \phi(S_t, A_t) \,\middle|\, S_0 = s \right] \tag{9}
$$

where $A_t \sim \pi(\cdot \mid S_t)$. Then, for any initial state distribution $\mu_0$ and $r \in \mathcal{F}$, we have $J(\pi, r) = \mathbb{E}_{s_0 \sim \mu}[\langle \psi^\pi(s_0), w_r \rangle]$, where $r(x, a) = \langle \phi(x, a), w_r \rangle$ — this holds by the linearity of expectation.

Now, we define $\Psi = \{ \mathbb{E}_{s_0 \sim \mu} \psi^\pi(s_0) : \pi \in \Pi \} \subset \mathbf{R}^d$. Note that $s \mapsto \psi_i^\pi(s)$ is equivalent to the value function under policy $\pi$ for the reward function $\phi_i$, for any $i \in [d]$. Thus, since $\Pi$ is assumed to be compact and $\phi$ is bounded, Dadashi et al. [2019] shows that the set $\mathcal{V}_i = \{ \psi_i^\pi : \pi \in \Pi \}$ is convex and compact for each $i \in [d]$. By continuity, $\overline{\mathcal{V}_i} = \{ \mathbb{E}_{s_0 \in \mu} \psi_i^\pi(s_0) : \pi \in \Pi \}$ is convex compact. Therefore, it holds that as a product of convex and compact sets, $\Psi = \bigotimes_{i=1}^d \overline{\mathcal{V}_i}$ is convex and compact.

Next, suppose $(\pi^\star, c^\star - r)$ realizes $\mathsf{Opt}_{\mathsf{IRL}}(\Pi, \mathcal{F} - r)$. Since $\Psi$ is compact and convex and $\mathcal{F}$ is convex, we apply Sion's minimax theorem and observe that, for $\psi_E := \mathbb{E}_{s_0 \sim \mu_0} \psi^{\pi_E}(s_0)$,

$$
\begin{aligned}
J(\pi^\star, c^\star - r) - J(\pi_E, c^\star - r) &= \mathsf{Opt}_{\mathsf{IRL}}(\Pi, \mathcal{F} - r) \\
&= \min_{\pi \in \Pi} \sup_{c \in \mathcal{F}} J(\pi, c - r) - J(\pi_E, c - r) \\
&= \min_{\psi \in \Psi} \sup_{w_c \in \mathbb{R}^d} \langle \psi - \psi_E, w_c - w_r \rangle \\
&= \min_{\psi \in \Psi} \sup_{w_c \in \mathbb{R}^d} \langle \psi_E - \psi, w_r - w_c \rangle \\
&= \sup_{w_c \in \mathbb{R}^d} \min_{\psi \in \Psi} \langle \psi_E - \psi, w_r - w_c \rangle \qquad \text{Sion's minimax theorem} \\
&= \sup_{c \in \mathcal{F}} \min_{\pi \in \Pi} J(\pi_E, r - c) - J(\pi, r - c) \\
&= \mathsf{Opt}_{\mathsf{S-ICRL}}(\Pi, \mathcal{F}, \mathcal{F}) \\
&= \mathsf{Opt}_{\mathsf{ICRL}}(\Pi, \mathcal{F}, \mathcal{F}) \qquad\qquad \text{Step 1} \\
\therefore J(\pi_E, r - c^\star) - J(\pi^\star, r - c^\star) &= \mathsf{Opt}_{\mathsf{ICRL}}(\Pi, \mathcal{F}, \mathcal{F}),
\end{aligned}
$$

where the penultimate step holds because $\mathcal{F}$, as a vector space, is a convex cone. Thus, we conclude that solving the IRL problem over the reward class $\mathcal{F} - r$ and policy class $\Pi$ solves the inverse constrained RL problem of interest. ∎

## C  Additional Results

### C.1  ICRL SAC Implementation Results

We attempted to reproduce the results of Liu et al. [2023a] using SAC for policy learning. We tested six settings for the Lagrangian and reward learning rates shown in Figure 5. All other hyperparameters were kept the same as Appendix A.1, except we ran each configuration for 3M timesteps. Notably, we had considerable difficulty reproducing the results of Liu et al. [2023a] using the SAC algorithm. While Kim et al. [2023] provided a SAC implementation of a similar algorithm, we note that their implementation added various additional components to the algorithm to improve convergence, including in particular, interleaving reward learning with behavior cloning. We did not use these techniques here.

### C.2  Learning Curves

Here we provide the full learning curves for all ablations and all environments. An additional configuration, **IRL-Base+BN** is included which was tested but not included in the main results due to poor performance. This configuration added batch normalization to the output of the reward function to potentially help combat issues with vanishing gradients.

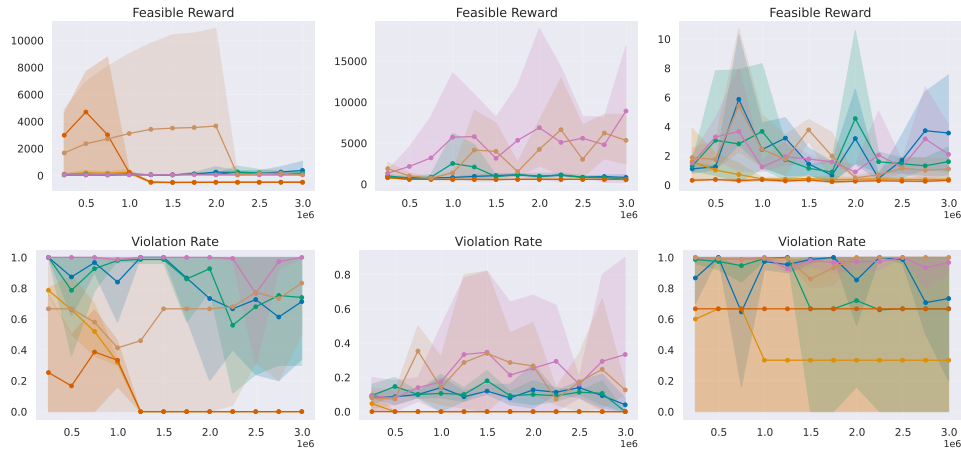

(a) Half Cheetah       (b) Ant       (c) Inverted Pendulum

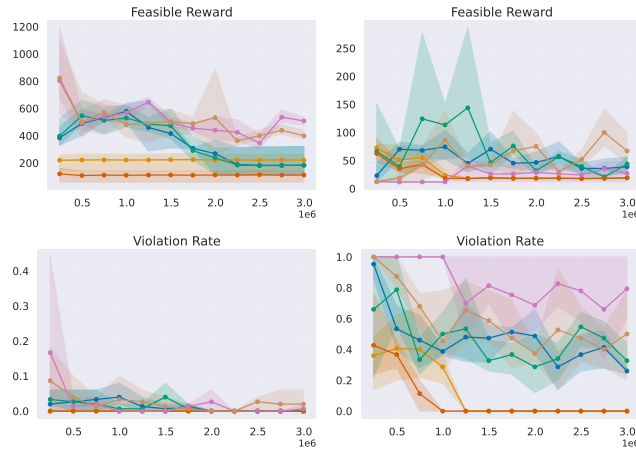

(d) Walker2D       (e) Swimmer

Figure 5: Training curves for feasible rewards (top) and violation rate (below) for ICRL with SAC with varying learning rates for the reward function and Lagrange multiplier across the five MuJoCo environments. None of the hyperparameter settings perform consistently well across environments.

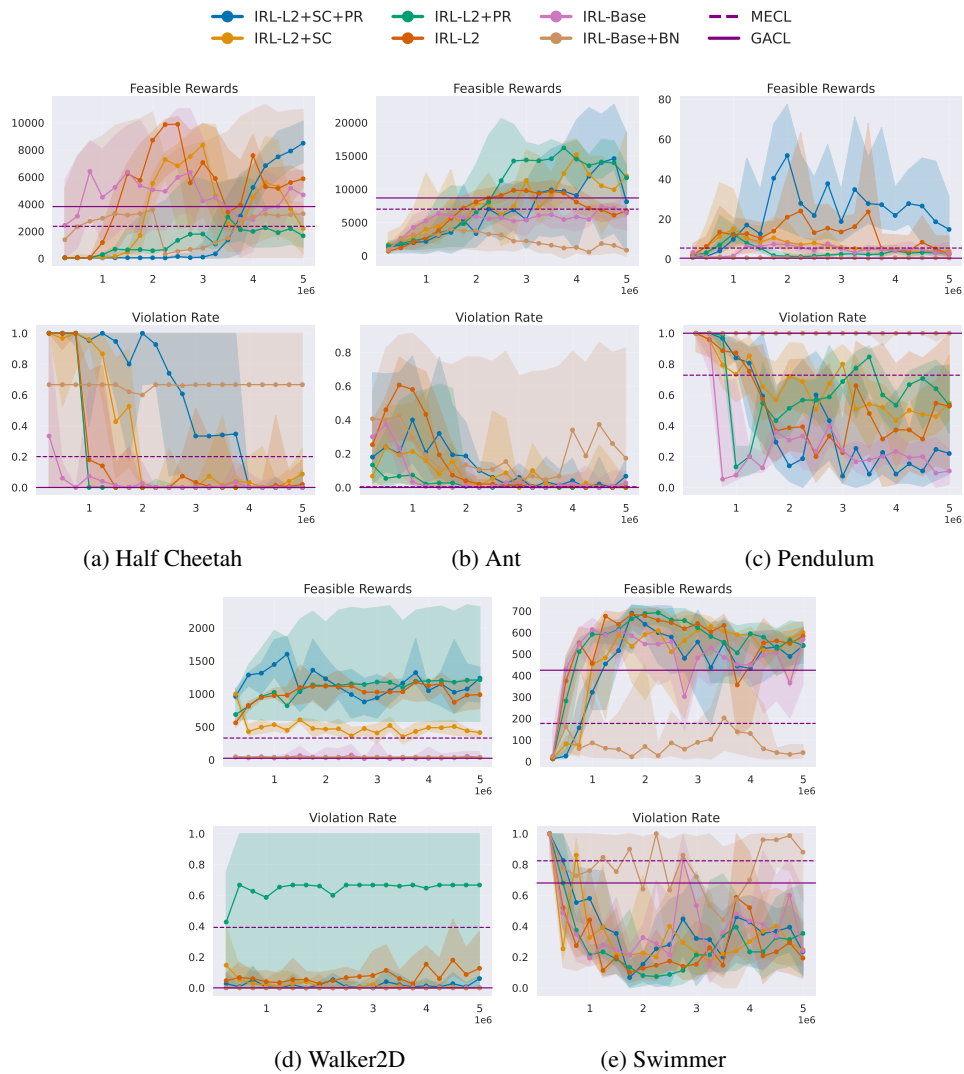

Figure 6: Training curves for feasible rewards (top) and violation rate (below) for all variations across the five MuJoCo environments.

